# Statistical Modeling of Images with Fields of Gaussian Scale Mixtures

**Siwei Lyu**    **Eero. P. Simoncelli**
Howard Hughes Medical Institute
Center for Neural Science, and
Courant Institute of Mathematical Sciences
New York University, New York, NY 10003

## Abstract

The local statistical properties of photographic images, when represented in a multi-scale basis, have been described using Gaussian scale mixtures (GSMs). Here, we use this local description to construct a global field of Gaussian scale mixtures (FoGSM). Specifically, we model subbands of wavelet coefficients as a product of an exponentiated homogeneous Gaussian Markov random field (hGMRF) and a second independent hGMRF. We show that parameter estimation for FoGSM is feasible, and that samples drawn from an estimated FoGSM model have marginal and joint statistics similar to wavelet coefficients of photographic images. We develop an algorithm for image denoising based on the FoGSM model, and demonstrate substantial improvements over current state-of-the-art denoising method based on the local GSM model.

Many successful methods in image processing and computer vision rely on statistical models for images, and it is thus of continuing interest to develop improved models, both in terms of their ability to precisely capture image structures, and in terms of their tractability when used in applications. Constructing such a model is difficult, primarily because of the intrinsic high dimensionality of the space of images. Two simplifying assumptions are usually made to reduce model complexity. The first is *Markovianity*: the density of a pixel conditioned on a small neighborhood, is assumed to be independent from the rest of the image. The second assumption is *homogeneity*: the local density is assumed to be independent of its absolute position within the image. The set of models satisfying both of these assumptions constitute the class of homogeneous Markov random fields (hMRFs).

Over the past two decades, studies of photographic images represented with multi-scale multi-orientation image decompositions (loosely referred to as "wavelets") have revealed striking non-Gaussian regularities and inter and intra-subband dependencies. For instance, wavelet coefficients generally have highly kurtotic marginal distributions [1, 2], and their amplitudes exhibit strong correlations with the amplitudes of nearby coefficients [3, 4]. One model that can capture the non-Gaussian marginal behaviors is a product of non-Gaussian scalar variables [5]. A number of authors have developed non-Gaussian MRF models based on this sort of local description [6, 7, 8], among which the recently developed *fields of experts* model [7] has demonstrated impressive performance in denoising (albeit at an extremely high computational cost in learning model parameters).

An alternative model that can capture non-Gaussian local structure is a scale mixture model [9, 10, 11]. An important special case is Gaussian scale mixtures (GSM), which consists of a Gaussian random vector whose amplitude is modulated by a hidden scaling variable. The GSM model provides a particularly good description of local image statistics, and the Gaussian substructure of the model leads to efficient algorithms for parameter estimation and inference. Local GSM-based methods represent the current state-of-the-art in image denoising [12]. The power of GSM models should be substantially improved when extended to describe more than a small neighborhood of wavelet coefficients. To this end, several authors have embedded local Gaussian mixtures into tree-structured

MRF models [e.g., 13, 14]. In order to maintain tractability, these models are arranged such that coefficients are grouped in non-overlapping clusters, allowing a graphical probability model with no loops. Despite their global consistency, the artificially imposed cluster boundaries lead to substantial artifacts in applications such as denoising.

In this paper, we use a local GSM as a basis for a globally consistent and spatially homogeneous field of Gaussian scale mixtures (FoGSM). Specifically, the FoGSM is formulated as the product of two mutually independent MRFs: a positive multiplier field obtained by exponentiating a homogeneous Gaussian MRF (hGMRF), and a second hGMRF. We develop a parameter estimation procedure, and show that the model is able to capture important statistical regularities in the marginal and joint wavelet statistics of a photographic image. We apply the FoGSM to image denoising, demonstrating substantial improvement over the previous state-of-the-art results obtained with a local GSM model.

# 1 Gaussian scale mixtures

A GSM random vector $\mathbf{x}$ is formed as the product of a zero-mean Gaussian random vector $\mathbf{u}$ and an independent random variable $z$, as $\mathbf{x} \overset{d}{=} \sqrt{z}\mathbf{u}$, where $\overset{d}{=}$ denotes equality in distribution. The density of $\mathbf{x}$ is determined by the covariance of the Gaussian vector, $\Sigma$, and the density of the multiplier, $p_z(z)$, through the integral

$$p(\mathbf{x}) = \int_z \mathcal{N}_{\mathbf{x}}(0, z\Sigma)p_z(z)dz \propto \int_z \frac{1}{\sqrt{z|\Sigma|}} \exp\left(-\frac{\mathbf{x}^T\Sigma^{-1}\mathbf{x}}{2z}\right)p_z(z)dz. \tag{1}$$

A key property of GSMs is that when $z$ determines the scale of the conditional variance of $\mathbf{x}$ given $z$, which is a Gaussian variable with zero mean and covariance $z\Sigma$. In addition, the normalized variable $\mathbf{x}\sqrt{z}$ is a zero mean Gaussian with covariance matrix $\Sigma$.

The GSM model has been used to describe the marginal and joint densities of local clusters of wavelet coefficients, both within and across subbands [9], where the embedded Gaussian structure affords simple and efficient computation. This local GSM model has been be used for denoising, by independently estimating each coefficient conditioned on its surrounding cluster [12]. This method achieves state-of-the-art performances, despite the fact that treating overlapping clusters as independent does not give rise to a globally consistent statistical model that satisfies all the local constraints.

# 2 Fields of Gaussian scale mixtures

In this section, we develop fields of Gaussian scale mixtures (FoGSM) as a framework for modeling wavelet coefficients of photographic images. Analogous to the local GSM model, we use a latent multiplier field to modulate a homogeneous Gaussian MRF (hGMRF). Formally, we define a FoGSM $\mathbf{x}$ as the product of two mutually independent MRFs,

$$\mathbf{x} \overset{d}{=} \mathbf{u} \otimes \sqrt{\mathbf{z}}, \tag{2}$$

where $\mathbf{u}$ is a zero-mean hGMRF, and $\mathbf{z}$ is a field of positive multipliers that control the local coefficient variances. The operator $\otimes$ denotes element-wise multiplication, and the square root operation is applied to each component. Note that $\mathbf{x}$ has a one-dimensional GSM marginal distributions, while its components have dependencies captured by the MRF structures of $\mathbf{u}$ and $\mathbf{z}$.

Analogous to the local GSM, when conditioned on $\mathbf{z}$, $\mathbf{x}$ is an *inhomogeneous* GMRF

$$p(\mathbf{x}|\mathbf{z}) \propto \sqrt{\frac{|Q_u|}{\prod_i z_i}} \exp\left(-\frac{1}{2}\mathbf{x}^T\left[\mathcal{D}\left(\sqrt{\mathbf{z}}\right)^{-1}Q_u\,\mathcal{D}\left(\sqrt{\mathbf{z}}\right)^{-1}\right]\mathbf{x}\right) = \sqrt{\frac{|Q_u|}{\prod_i z_i}} \exp\left(-\frac{1}{2}(\mathbf{x} \oslash \sqrt{\mathbf{z}})^T Q_u(\mathbf{x} \oslash \sqrt{\mathbf{z}})\right), \tag{3}$$

where $Q_u$ is the inverse covariance matrix of $\mathbf{u}$ (also known as the *precision matrix*), and $\mathcal{D}(\cdot)$ denotes the operator that form a diagonal matrix from an input vector. Note also that the element-wise division of the two fields, $\mathbf{x} \oslash \sqrt{\mathbf{z}}$, yields a hGMRF with precision matrix $Q_u$.

To complete the FoGSM model, we need to specify the structure of the multiplier field $\mathbf{z}$. For tractability, we use another hGMRF as a substrate, and map it into positive values by exponentiation,

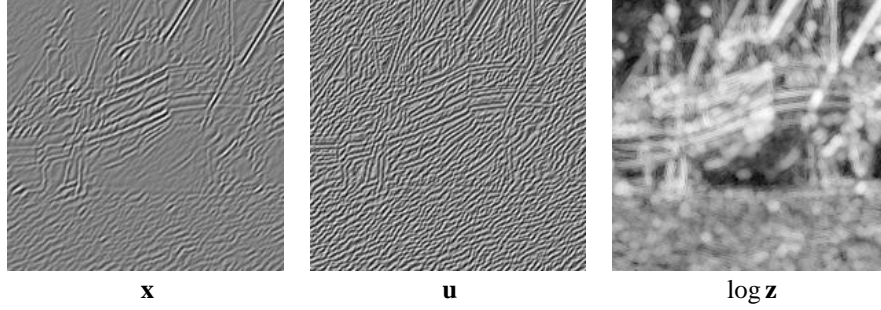

| **x** | **u** | $\log \mathbf{z}$ |

**Fig. 1.** Decomposition of a subband from image "boat" (left) into the normalized subband **u** (middle) and the multiplier field **z** (right, in the logarithm domain). Each image is rescaled individually to fill the full range of grayscale intensities.

as was done in [10]. To be more specific, we model $\log(\mathbf{z})$ as a hGMRF with mean $\mu$ and precision matrix $Q_z$, where the log operator is applied element-wise, from which the density of **z** follows as:

$$p_{\mathbf{z}}(\mathbf{z}) \propto \frac{\sqrt{|Q_z|}}{\prod_i z_i} \exp\left(-\frac{1}{2}(\log \mathbf{z} - \mu)^T Q_z (\log \mathbf{z} - \mu)\right). \tag{4}$$

This is a natural extension of the univariate lognormal prior used previously for the scalar multiplier in the local GSM model [12].

The restriction to hGMRFs greatly simplifies computation with FoGSM. Particularly, we take advantage of the fact that a 2D hGMRF with circular boundary handling has a sparse block-circulant precision matrix with a generating kernel $\theta$ specifying its nonzero elements. A block-circulant matrix is diagonalized by the Fourier transform, and its multiplication with a vector corresponds to convolution with the kernel $\theta$. The diagonalizability with a fixed and efficiently computed transform makes the parameter estimation, sampling, and inference with a hGMRF substantially more tractable than with a general MRF. Readers are referred to [15] for a detailed description of hGMRFs.

**Parameter estimation**: The estimation of the latent multiplier field **z** and the model parameters $(\mu, Q_z, Q_u)$ may be achieved by maximizing $\log p(\mathbf{x}, \mathbf{z}; Q_u, Q_z, \mu)$ with an iterative coordinate-ascent method, which is guaranteed to converge. Specifically, based on the statistical dependency structures in the FoGSM model, the following three steps are repeated until convergence:

$$
\begin{array}{rll}
(i) & \mathbf{z}^{(t+1)} = & \operatorname{argmax}_{\mathbf{z}} \log p(\mathbf{x}|\mathbf{z}; Q_u^{(t)}) + \log p(\mathbf{z}; Q_z^{(t)}, \mu^{(t)}) \\
(ii) & Q_u^{(t+1)} = & \operatorname{argmax}_{Q_u} \log p(\mathbf{x}|\mathbf{z}^{(t+1)}; Q_u) \\
(iii) & (Q_z^{(t+1)}, \mu^{(t+1)}) = & \operatorname{argmax}_{Q_z, \mu} \log p(\mathbf{z}^{(t+1)}; Q_z, \mu)
\end{array} \tag{5}
$$

According to the FoGSM model structure, steps (ii) and (iii) correspond to maximum likelihood estimates of the parameters of hGMRFs, $\left[\mathbf{x} \oslash \sqrt{\mathbf{z}^{(t+1)}}\right]$ and $[\log \mathbf{z}^{(t+1)}]$, respectively. Because of this, both steps may be efficiently implemented by exploiting the diagonalization of the precision matrices with 2D Fourier transforms [15].

Step (i) in (5) may be implemented with conjugate gradient ascent [16]. To simplify description and computation, we introduce a new variable for the element-wise inverse square root of the multiplier: $\mathbf{s} = 1 \oslash \sqrt{\mathbf{z}}$. The likelihood in (3) is then changed to:

$$p(\mathbf{x}|\mathbf{s}) \propto \prod_i s_i \exp\left(-\frac{1}{2}(\mathbf{x} \otimes \mathbf{s})^T Q_u (\mathbf{x} \otimes \mathbf{s})\right) = \prod_i s_i \exp\left(-\frac{1}{2}\mathbf{s}^T \mathcal{D}(\mathbf{x}) Q_u \mathcal{D}(\mathbf{x}) \mathbf{s}\right). \tag{6}$$

The joint density of **s** is obtained from (4), using the relations between densities of transformed variables, as

$$p(\mathbf{s}) \propto \frac{1}{\prod_i s_i} \exp\left(-\frac{1}{2}(2\log \mathbf{s} + \mu)^T Q_z (2\log \mathbf{s} + \mu)\right). \tag{7}$$

Combining . (6) and (7), step (i) in (5) is equivalent to computing $\hat{\mathbf{s}} \overset{\triangle}{=} \operatorname{argmax}_{\mathbf{s}} \log p(\mathbf{x}|\mathbf{s}; Q_u) + \log p(\mathbf{s}; Q_z, \mu)$, which is further simplified into:

$$\operatorname*{argmin}_{\mathbf{s}} \left\{ \frac{1}{2}\mathbf{s}^T \mathcal{D}(\mathbf{x}) Q_u \mathcal{D}(\mathbf{x}) \mathbf{s} + \frac{1}{2}(2\log \mathbf{s} + \mu)^T Q_z (2\log \mathbf{s} + \mu) \right\}. \tag{8}$$

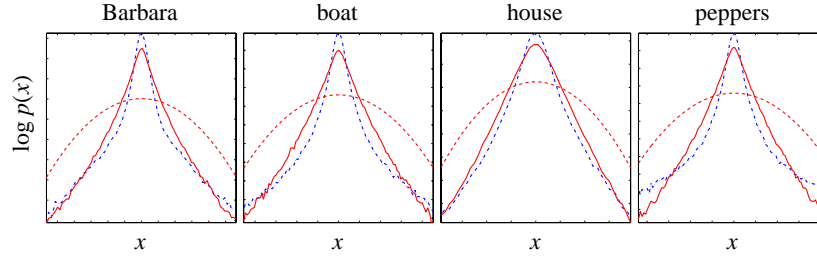

**Fig. 2.** Empirical marginal log distributions of coefficients from a multi-scale decomposition of photographic images (blue dot-dashed line), synthesized FoGSM samples from the same subband (red solid line), and a Gaussian with the same standard deviation (red dashed line).

and the optimal $\hat{\mathbf{z}}$ is then recovered as $\hat{\mathbf{z}} = 1 \oslash (\hat{\mathbf{s}} \otimes \hat{\mathbf{s}})$. We the optimize (8) with conjugate gradient ascent [16]. Specifically, the negative gradient of the objective function in (8) with respect to $\mathbf{s}$ is

$$-\frac{\partial \log p(\mathbf{x}|\mathbf{s})p(\mathbf{s})}{\partial \mathbf{s}} = \mathcal{D}(\mathbf{x})\,Q_u\,\mathcal{D}(\mathbf{x})\,\mathbf{s} + 2\,\mathcal{D}(\mathbf{s})^{-1}Q_z(2\log \mathbf{s} + \mu)$$

$$= \mathbf{x} \otimes (\theta_u \star (\mathbf{x} \otimes \mathbf{s})) + 2(\theta_z \star (2\log \mathbf{s} + \mu)) \oslash \mathbf{s},$$

and the multiplication of any vector $\mathbf{h}$ with the Hessian matrix can be computed as:

$$\frac{\partial^2 \log p(\mathbf{x}|\mathbf{s})p(\mathbf{s})}{\partial \mathbf{s}^2}\mathbf{h} = \mathbf{x} \otimes (\theta_u \star (\mathbf{x} \otimes \mathbf{h})) + 4\,(\theta_z \star (\mathbf{h} \oslash \mathbf{s})) \oslash \mathbf{s} - 2\,(\theta_z \star (\log \mathbf{s} + \mu)) \otimes \mathbf{h} \oslash (\mathbf{s} \otimes \mathbf{s}).$$

Both operations can be expressed entirely in terms of element-wise operations ($\oslash$ and $\otimes$) and 2D convolutions ($\star$) with the generating kernels of the two precision matrices $\theta_u$ and $\theta_z$, which allows for efficient implementation.

## 3   Modeling photographic images

We have applied the FoGSM model to subbands of a multi-scale image representation known as a steerable pyramid [17]. This decomposition is a tight frame, constructed from oriented multi-scale derivative operators, and is overcomplete by a factor of $4K/3$, where $K$ is the number of orientation bands. Note that the marginal and joint statistics we describe are not specific to this decomposition, and are similar for other multi-scale oriented representations. We fit a FoGSM model to each subband of a decomposed photographic image, using the algorithms described in the previous section. For precision matrices $Q_u$ and $Q_z$, we assumed a $5 \times 5$ Markov neighborhood (corresponding to a $5 \times 5$ convolution kernel), which was loosely chosen to optimize the tradeoff between accuracy and overfitting.

Figure 1 shows the result of fitting a FoGSM model to an example subband from the "boat" image (left panel). The subband is decomposed into the product of the $\mathbf{u}$ field (middle panel) and the $\mathbf{z}$ field (right panel, in the logarithm domain), along with model parameters $Q_u$, $\mu$ and $Q_z$ (not shown). Visually, the changing spatial variances are represented in the estimated $\log \mathbf{z}$ field, and the estimated $\mathbf{u}$ is much more homogeneous than the original subband and has a marginal distribution close to Gaussian.[1]  However, the $\log \mathbf{z}$ field still has a non-Gaussian marginal distribution and is spatially inhomogeneous, suggesting limitations of FoGSM for modeling photographic image wavelet coefficients (see Discussion).

The statistical dependencies captured by the FoGSM model can be further revealed by examining marginal and joint statistics of samples synthesized with the estimated model parameters. A sample from FoGSM can be formed by multiplying samples of $\mathbf{u}$ and $\sqrt{\mathbf{z}}$. The former is obtained by sampling from hGMRF $\mathbf{u}$, and the latter is obtained from the element-wise exponentiation followed by a element-wise square root operation of a sample of hGMRF $\log \mathbf{z}$. This procedure is again efficient for FoGSM due to the use of hGMRFs as building blocks [15].

**Marginal distributions**: We start by comparing the marginal distributions of the samples and the original subband. Figure 2 shows empirical histograms in the log domain of a particular subband

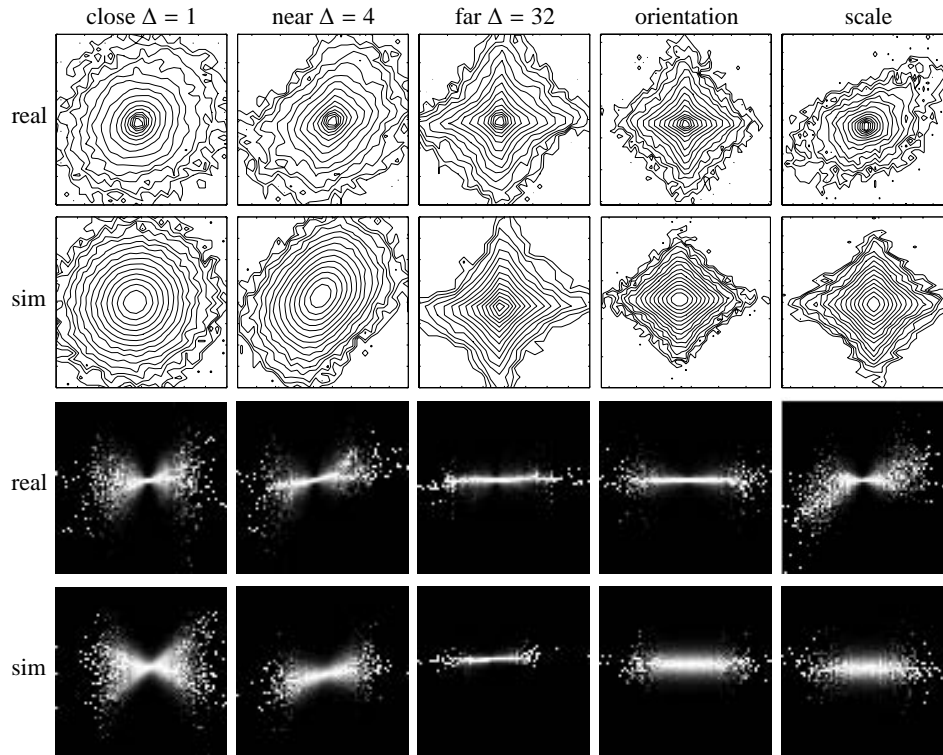

**Fig. 3.** Examples of empirically observed distributions of wavelet coefficient pairs, compared with distributions from synthesized samples with the FoGSM model. See text for details.

from four different photographic images (blue dot-dashed line), and those of the synthesized samples of FoGSM models learned from each corresponding subband (red solid line). For comparison, a Gaussian with the same standard deviation as the image subband is also displayed (red dashed line). Note that the synthesized samples have conspicuous non-Gaussian characteristics similar to the real subbands, exemplified by the high peak and heavy tails in the marginal distributions. On the other hand, they are typically less kurtotic than the real subbands. We believe this arises from the imprecise Gaussian approximation of log $\mathbf{z}$ (see Discussion).

**Joint distributions**: In addition to one-dimensional marginal statistics, the FoGSM model is capable of capturing the joint behavior of wavelet coefficients. As described in [4, 9], wavelet coefficients of photographic images present non-Gaussian dependencies. Shown in the first and the third rows in Fig. 3 are empirical joint and conditional histograms for one subband of the "boat" image, for five pairs of coefficients, corresponding to basis functions with spatial separations of $\Delta = \{1, 4, 32\}$ samples, two orthogonal orientations and two adjacent scales. Contour lines in the joint histogram are drawn at equal intervals of log probability. Intensities in the conditional histograms correspond to probability, except that each column is independently rescaled to fill the full range of intensity. For a pair of adjacent coefficients, we observe an elliptical joint distribution and a "bow-tie" shaped conditional distribution. The latter is indicative of strong non-Gaussian dependencies. For coefficients that are distant, the dependency becomes weaker and the corresponding joint and conditional histograms become more separable, as would be expected for two independent random variables.

Random samples drawn from a FoGSM model, with parameters fitted to the corresponding subband, have statistical characteristics consistent with the general description of wavelet coefficients of photographic images. Shown in the second and the fourth rows of Fig. 3 are the joint and conditional histograms of synthesized samples from the FoGSM model estimated from the same subband as in the first and the third rows. Note that the joint and conditional histograms of the synthesized samples have similar transition of spatial dependencies as the separation increases (column 1,2 and 3), suggesting that the FoGSM accounts well for pairwise joint dependencies of coefficients over a full range of spatial separations. On the other hand, the dependencies between subbands of different orientations and scales are not properly modeled by FoGSM (column 4 and 5). This is especially true for subbands at different scales, which exhibit strong dependencies. The current FoGSM model

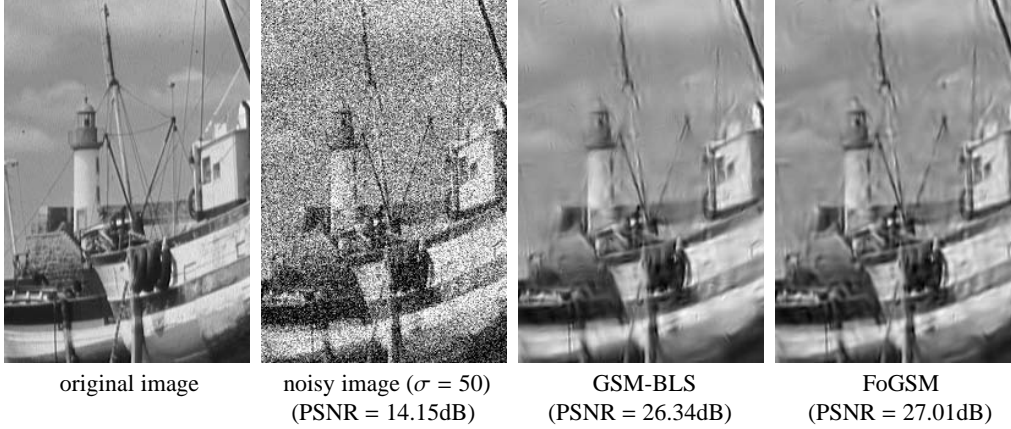

| original image | noisy image ($\sigma$ = 50) | GSM-BLS | FoGSM |
|---|---|---|---|
| | (PSNR = 14.15dB) | (PSNR = 26.34dB) | (PSNR = 27.01dB) |

**Fig. 4.** Denoising results using local GSM [12] and FoGSM. Performances are evaluated in peak-signal-to-noise-ratio (PSNR), $20 \log_{10}(255/\sigma_e)$, where $\sigma_e$ is the standard deviation of the error.

does not exhibit those dependencies as only spatial neighbors are used to make use of the 2D hGMRFs (see Discussion).

## 4 Application to image denoising

Let $\mathbf{y} = \mathbf{x} + \mathbf{w}$ be a wavelet subband of an image that has been corrupted with white Gaussian noise of known variance. In an overcomplete wavelet domain such as steerable pyramid, the white Gaussian noise is transformed into correlated Gaussian noise $\mathbf{w} \sim \mathcal{N}_{\mathbf{w}}(0, \Sigma_w)$, whose covariance $\Sigma_w$ can be derived from the basis functions of the pyramid transform. With FoGSM as prior over $\mathbf{x}$, commonly used denoising methods involve expensive high-dimensional integration: for instance, *maximum a posterior* estimate, $\hat{\mathbf{x}}_{MAP} = \text{argmax}_{\mathbf{x}} \log p(\mathbf{x}|\mathbf{y})$, requires a high-dimensional integral over $\mathbf{z}$, and the Bayesian least square estimation, $\hat{\mathbf{x}}_{BLS} = E(\mathbf{x}|\mathbf{y})$ requires a double high-dimensional integral over $\mathbf{x}$ and $\mathbf{z}$. Although it is possible to optimize with these criteria using Monte-Carlo Markov sampling or other approximations, we instead develop a more efficient deterministic algorithm that takes advantage of the hGMRF structure in the FoGSM model. Specifically, we compute

$$(\hat{\mathbf{x}}, \hat{\mathbf{z}}, \hat{Q}_u, \hat{Q}_z, \hat{\mu}) = \text{argmax}_{\mathbf{x},\mathbf{z},Q_u,Q_z,\mu} \log p(\mathbf{x}, \mathbf{z}|\mathbf{y}; Q_u, Q_z, \mu) \tag{9}$$

and take $\hat{\mathbf{x}}$ as the optimal denoised subband. Note that the model parameters are learned within the inference process rather than in a separate parameter learning step. This strategy, known as partial optimal solution [19], greatly reduces the computational complexity.

We optimize (9) with coordinate ascent, iterating between maximizing each of $(\mathbf{x}, \mathbf{z}, Q_u, Q_z, \mu)$ while fixing the others. With fixed estimates of $(\mathbf{z}, Q_u, Q_z, \mu)$, the optimization of $\mathbf{x}$ is

$$\text{argmax}_{\mathbf{x}} \log p(\mathbf{x}, \mathbf{z}|\mathbf{y}; Q_u, Q_z, \mu) = \text{argmax}_{\mathbf{x}} \{\log p(\mathbf{x}|\mathbf{z}, \mathbf{y}; Q_u, Q_z, \mu) + \log p(\mathbf{z}|\mathbf{y}; Q_u, Q_z, \mu)\},$$

which reduces to $\text{argmax}_{\mathbf{x}} \log p(\mathbf{x}|\mathbf{z}, \mathbf{y}; Q_u)$, with the second term independent of $\mathbf{x}$ and can be dropped from optimization. Given the Gaussian structure of $\mathbf{x}$ given $\mathbf{z}$, this step is then equivalent to a Wiener filter (linear in $\mathbf{y}$). Fixing $(\mathbf{x}, Q_u, Q_z, \mu)$, the optimization of $\mathbf{z}$ is

$$\text{argmax}_{\mathbf{z}} \log p(\mathbf{x}, \mathbf{z}|\mathbf{y}; Q_u, Q_z, \mu) = \text{argmax}_{\mathbf{z}} \{\log p(\mathbf{y}|\mathbf{x}, \mathbf{z}; Q_u) + \log p(\mathbf{x}, \mathbf{z}; Q_u, Q_z, \mu) - \log p(\mathbf{y}; Q_u, Q_z, \mu)\},$$

which is further reduced to $\text{argmax}_{\mathbf{z}} \log p(\mathbf{x}, \mathbf{z}; Q_u, Q_z, \mu)$. Here, the first term was dropped since $\mathbf{y}$ is independent of $\mathbf{z}$ when conditioned on $\mathbf{x}$. The last term was also dropped since it is also independent of $\mathbf{z}$. Therefore, optimizing $\mathbf{z}$ given $(\mathbf{x}, Q_u, Q_z, \mu)$ is equivalent to the first step of the algorithm in section 2, which can be implemented with efficient gradient descent. Finally, given $(\mathbf{x}, \mathbf{z})$, the FoGSM model parameters $(Q_u, Q_z, \mu)$ are estimated from hGMRFs $\left[\mathbf{x} \oslash \sqrt{\mathbf{z}^{(t+1)}}\right]$ and $[\log \mathbf{z}^{(t+1)}]$, similar to the second and third step in the algorithm of section 2. However, to reduce the overall computation time, instead of a complete maximum likelihood estimation, these parameters are estimated with a maximum pseudo-likelihood procedure [20], which finds the parameters maximizing the product of all conditional distributions (which are 1D Gaussians in the GMRF case), followed by a projection to the subspace of FoGSM parameters that results in positive definite precision matrices.

We tested this denoising method on a standard set of test images [12]. The noise corrupted images were first decomposed these into a steerable pyramid with multiple levels (5 levels for a $512 \times 512$ image and 4 levels for a $256 \times 256$ image ) and 8 orientations. We assumed a FoGSM model for each subband, with a $5 \times 5$ neighborhood for field $\mathbf{u}$ and a $1 \times 1$ neighborhood for field $\log \mathbf{z}$. These sizes were chosen to provide a reasonable combination of performance and computational efficiency. We then estimate the optimal $\mathbf{x}$ with the algorithm described previously, with the initial values of $\mathbf{x}$ and $\mathbf{z}$ computed from subband denoised with the local GSM model [12]. Shown in Fig. 4 is an example of denoising the "boat" image corrupted with simulated additive white Gaussian noise of strength $\sigma = 50$, corresponding to a peak-signal-to-noise-ratio (PSNR), of 14.15 dB. We compare this with the local GSM method in [12], which, assuming a local GSM model for the neighborhood consisting of $3 \times 3$ spatial neighbors plus parent in the next coarsest scale, computes a Bayes least squares estimate of each coefficient conditioned on its surrounding neighborhood. The FoGSM denoising achieves substantial improvement (+0.68 in PSNR) and is seen to exhibit better contrast and continuation of oriented features (see Fig. 4). On the other hand, FoGSM introduces some noticeable artifacts in low contrast areas, which is caused by numerical instability at locations with small $z$. We find that the improvement in terms of PSNR is consistent across photographic images and noise levels, as reported in Table 1. But even with a restricted neighborhood for the multiplier field, this PSNR improvement does come at a substantial computational cost. As a rough indication, running on a PowerPC G5 workstation with 2.3 Ghz processor and 16 Gb RAM memory, using unoptimized MATLAB (version R14) code, denoising a $512 \times 512$ image takes on average 4.5 hours (results averaging over 5 images), and denoising a $256 \times 256$ image takes on average 2.4 hours (result averaging over 2 images), to a convergence precision producing the reported results. Our preliminary investigation indicates that the slow running time is mainly due to the nature of coordinate ascent and the landscape of (9), which requires many iterations to converge.

## 5 Discussion

We have introduced fields of Gaussian scale mixtures as a flexible and efficient tool for modeling the statistics of wavelet coefficients of photographic images. We developed a feasible (although admittedly computationally costly) parameter estimation method, and showed that samples synthesized from the fitted FoGSM model are able to capture structures in the marginal and joint wavelet statistics of photographic images. Preliminary results of applying FoGSM to image denoising indicate substantial improvements over the state-of-the-art methods based on the local GSM model.

Although FoGSM has a structure that is similar to the local scale mixture model [9, 10], there is a fundamental difference between them. In FoGSM, hGMRF structures are enforced in $\mathbf{u}$ and $\log \mathbf{z}$, while the local scale mixture models impose minimal statistical structure on these variables. Because of this, our model easily extends to images of arbitrary size, while the local scale mixture models are essentially confined to describing small image patches (the curse of dimensionality, and the increase in computational cost prevent one from scaling the patch size up). On the other hand, the close relation to Gaussian MRF makes the analysis and computation of FoGSM significantly easier than other non-Gaussian MRF based image models [6, 7, 5].

We envision, and are currently working on, a number of model improvements. First, the model should benefit from the introduction of more general Markov neighborhoods, including wavelet coefficients from subbands at other scales and orientations [4, 12], since the current model is clearly not accounting for these dependencies (see Fig. 3). Secondly, the log transformation used to derive the multiplier field from a hGMRF is somewhat ad hoc, and we believe that substitution of another nonlinear transformation (e.g., a power law [14]) might lead to a more accurate description of the image statistics. Thirdly, the current denoising method estimates model parameter during the process of denoising, which produces image adaptive model parameters. We are exploring the possibility of using a set of generic model parameter learned *a priori* on a large set of photographic images, so that a generic statistical model for all photographic images based on FoGSM can be built. Finally, there exist residual inhomogeneous structures in the $\log \mathbf{z}$ field (see Fig. 1) that can likely be captured by explicitly incorporating local orientation [21] or phase into the model. Finding tractable models and algorithms for handling such circular variables is challenging, but we believe their inclusion will result in substantial improvements in modeling and in denoising performance.

| $\sigma$/PSNR | Barbara | barco | boat | fingerprint |
|---|---|---|---|---|
| 10/28.13 | 35.01 (34.01) | 35.05 (34.42) | 34.12 (33.58) | 33.28 (32.45) |
| 25/20.17 | 30.10 (29.07) | 30.44 (29.73) | 30.03 (29.34) | 28.45 (27.44) |
| 50/14.15 | 26.40 (25.45) | 27.36 (26.63) | 27.01 (26.35) | 25.11 (24.13) |
| 100/8.13 | 23.01 (22.61) | 24.44 (23.84) | 24.20 (23.79) | 21.78 (21.21) |
| $\sigma$/PSNR | Flintstones | house | Lena | peppers |
| 10/28.13 | 32.47 (31.78) | 35.63 (35.27) | 35.94 (35.60) | 34.38 (33.73) |
| 25/20.17 | 28.29 (27.48) | 31.64 (31.32) | 32.11 (31.70) | 29.78 (29.18) |
| 50/14.15 | 24.82 (24.02) | 28.51 (28.23) | 29.12 (28.62) | 26.43 (25.93) |
| 100/8.13 | 21.24 (20.49) | 25.33 (25.31) | 26.12 (25.77) | 23.17 (22.80) |

**Table 1.** Denoising results with FoGSM on different images and different noise levels. Shown in the table are PSNRs ($20 \log_{10}(255/\sigma_e)$, where $\sigma_e$ is the standard deviation of the error) of the denoised images, and in the parenthesis are the PSNRs of the same images denoised with a local GSM model [12].

## Footnotes

[1]This "Gaussianizing" behavior was first noted in photographic images by Ruderman [18], who observed that image derivative measurements that were normalized by a local estimate of their standard deviation had approximately Gaussian marginal distributions.

## References

[1] P. J. Burt. Fast filter transforms for image processing. *Comp. Graph. Image Proc.*, 16:20–51, 1981.

[2] D. J. Field. Relations between the statistics of natural images and the response properties of cortical cells. *J. Opt. Soc. Am.*, 4(12):2379–2394, 1987.

[3] B. Wegmann and C. Zetzsche. Statistical dependencies between orientation filter outputs used in human vision based image code. In *Proc. Visual Comm. and Image Proc.*, volume 1360, pages 909–922, 1990.

[4] R. W. Buccigrossi and E. P. Simoncelli. Image compression via joint statistical characterization in the wavelet domain. *IEEE Trans. on Image Proc.*, 8(12):1688–1701, 1999.

[5] Y. W. Teh, M. Welling, S. Osindero, and G. E. Hinton. Energy-based models for sparse overcomplete representations. *J. of Machine Learning Res.*, 4:1235–1260, 2003.

[6] S. C. Zhu, Y. Wu, and D. Mumford. Filters, random fields and maximum entropy (FRAME): Towards a unified theory for texture modeling. *Int'l. J. Comp. Vis.*, 27(2):107–126, 1998.

[7] S. Roth and M. J. Black. Fields of experts: a framework for learning image priors. In *IEEE Conf. on Comp. Vis. and Pat. Rec.*, volume 2, pages 860–867, 2005.

[8] P. Gehler and M. Welling. Products of "edge-perts". In *Adv. in Neural Info. Proc. Systems (NIPS*05)*. MIT Press, 2006.

[9] M. J. Wainwright and E. P. Simoncelli. Scale mixtures of Gaussians and the statistics of natural images. In *Adv. Neural Info. Proc. Sys. (NIPS*99)*, volume 12, pages 855–861, May 2000.

[10] Y. Karklin and M. S. Lewicki. A hierarchical Bayesian model for learning non-linear statistical regularities in non-stationary natural signals. *Neural Computation*, 17(2):397–423, 2005.

[11] A. Hyvärinen, P. O. Hoyer, and M. Inki. Topographic ICA as a model of natural image statistics. In *the First IEEE Int'l. Workshop on Bio. Motivated Comp. Vis.*, London, UK, 2000.

[12] J. Portilla, V. Strela, M. J. Wainwright, and E. P. Simoncelli. Image denoising using scale mixtures of Gaussians in the wavelet domain. *IEEE Trans. on Image Proc.*, 12(11):1338–1351, 2003.

[13] J. Romberg, H. Choi, and R. G. Baraniuk. Bayesian tree-structured image modeling using wavelet domain hidden Markov models. *IEEE Trans. on Image Proc.*, 10(7):303–347, 2001.

[14] M. J. Wainwright, E. P. Simoncelli, and A. S. Willsky. Random cascades on wavelet trees and their use in modeling and analyzing natural imagery. *Appl. and Comp. Harm. Ana.*, 11(1):89–123, 2001.

[15] H. Rue and L. Held. *Gaussian Markov Random Fields: Theory And Applications*. Monographs on Statistics and Applied Probability. Chapman and Hall/CRC, 2005.

[16] W. H. Press, S. A. Teukolsky, W. T. Vetterling, and B. P. Flannery. *Numerical Recipes*. Cambridge, 2nd edition, 2002.

[17] E. P. Simoncelli and W. T. Freeman. The steerable pyramid: A flexible architecture for multi-scale derivative computation. In *IEEE Int'l. Conf. on Image Proc.*, volume 3, pages 444–447, 1995.

[18] D. Ruderman. The statistics of natural images. *Network : Comp. in Neural Sys.*, 5:598–605, 1994.

[19] M. Figueiredo and J. Leitão. Unsupervised image restoration and edge location using compound Gauss-Markov random fields and MDL principle. *IEEE Trans. on Image Proc.*, 6(8):1089–1122, 1997.

[20] J. Besag. On the statistical analysis of dirty pictures. *J. of the Royal Stat. Soc., Series B*, 48:259–302, 1986.

[21] D. K. Hammond and E. P. Simoncelli. Image denoising with an orientation-adaptive Gaussian scale mixture model. In *Proc. 13th IEEE Int'l. Conf. on Image Proc.*, pages 1433–1436, October 2006.
